# A Silicon Axon

**Bradley A. Minch, Paul Hasler, Chris Diorio, Carver Mead**

Physics of Computation Laboratory
California Institute of Technology
Pasadena, CA 91125

bminch, paul, chris, carver@pcmp.caltech.edu

## Abstract

We present a silicon model of an axon which shows promise as a building block for pulse-based neural computations involving correlations of pulses across both space and time. The circuit shares a number of features with its biological counterpart including an excitation threshold, a brief refractory period after pulse completion, pulse amplitude restoration, and pulse width restoration. We provide a simple explanation of circuit operation and present data from a chip fabricated in a standard $2\mu m$ CMOS process through the MOS Implementation Service (MOSIS). We emphasize the necessity of the restoration of the width of the pulse in time for stable propagation in axons.

## 1  INTRODUCTION

It is well known that axons are neural processes specialized for transmitting information over relatively long distances in the nervous system. Impulsive electrical disturbances known as *action potentials* are normally initiated near the cell body of a neuron when the voltage across the cell membrane crosses a threshold. These pulses are then propagated with a fairly stereotypical shape at a more or less constant velocity down the length of the axon. Consequently, axons excel at precisely preserving the relative timing of threshold crossing events but do not preserve any of the initial signal shape. Information, then, is presumably encoded in the relative timing of action potentials.

The biophysical mechanisms underlying the initiation and propagation of action potentials in axons have been well studied since the seminal work of Hodgkin and Huxley on the giant axon of *Loligo*. (Hodgkin & Huxley, 1952) Briefly, when the voltage across a small patch of the cell membrane increases to a certain level, a population of ion channels permeable to sodium opens, allowing an influx of sodium ions, which, in turn, causes the membrane voltage to increase further and a pulse to be initiated. This population of channels rapidly inactivates, preventing the passage of additional ions. Another population of channels permeable to potassium opens after a brief delay causing an efflux of potassium ions, restoring the membrane to a more negative potential and terminating the pulse. This cycle of ion migration is coupled to neighboring sections of the axon, causing the action potential to propagate. The sodium channels remain inactivated for a brief interval of time during which the affected patch of membrane will not be able to support another action potential. This period of time is known as the refractory period. The axon circuit which we present in this paper does not attempt to model the detailed dynamics of the various populations of ion channels, although such detailed neuromimes are both possible (Lewis, 1968; Mahowald & Douglas, 1991) and useful for learning about natural neural systems. Nonetheless, it shares a number of important features with its biological counterpart including having a threshold for excitation and a refractory period.

It is well accepted that the amplitude of the action potential must be restored as it propagates. It is not as universally understood is that the *width* of the action potential must be restored *in time* if it is to propagate over any appreciable distance. Otherwise, the pulse would smear out in time resulting in a loss of precise timing information, or it would shrink down to nothing and cease to propagate altogether. In biological axons, restoration of the pulse width is accomplished through the dynamics of sodium channel inactivation and potassium channel activation. In our silicon model, the pulse width is restored through feedback from the successive stage. This feedback provides an inactivation which is similar to that of the sodium channels in biological axons and is also the underlying cause of refractoriness in our circuit.

In the following section we provide a simple description of how the circuit behaves. Following this, data from a chip fabricated in a standard $2\mu m$ CMOS process through MOSIS are presented and discussed.

## 2   THE SILICON AXON CIRCUIT

An axon circuit which is to be used as a building block in large-scale computational systems should be made as simple and low-power as possible, since it would be replicated many times in any such system. Each stage of the axon circuit described below consists of five transistors and two small capacitors, making the axon circuit very compact. The axon circuit uses the delay through a stage to time the signal which is fed back to restore the pulse width, thus avoiding the need for an additional delay circuit for each section. Additonally, the circuit operates with low power; during typical operation (a pulse of width $2ms$ travelling at $10^3 stages/s$), pulse propagation costs about $4pJ/stage$ of energy. Under these circumstances, the circuit consumes about $2nW/stage$ of static power.

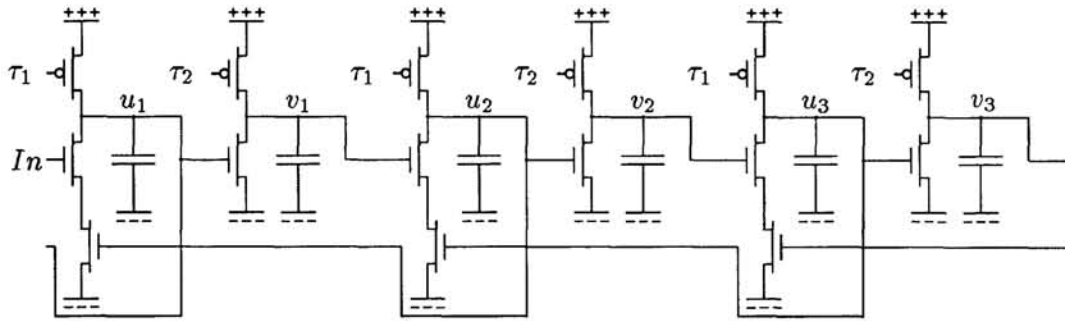

Figure 1: Three sections of the axon circuit.

Three stages of the axon circuit are depicted in Figure 1. A single stage consists of two capacitors and what would be considered a pseudo-nMOS NAND gate and a pseudo-nMOS inverter in digital logic design. These simple circuits are characterized by a *threshold voltage* for switching and a *slew rate* for recharging. Consider the inverter circuit. If the input is held low for a sufficiently long time, the pull-up transistor will have charged the output voltage almost completely to the positive rail. If the input voltage is ramped up toward the positive rail, the current in the pull-down transistor will increase rapidly. At some input voltage level, the current in the pull-down transistor will equal the saturation current of the pull-up transistor; this voltage is known as the threshold. The output voltage will begin to discharge at a rapidly increasing rate as the input voltage is increased further. After a very short time, the output will have discharged almost all the way to the negative rail. Now, if the input were decreased rapidly, the output voltage would ramp linearly in time (slew) up toward the positive rail at a rate set by the saturation current in the pull-up transistor and the capacitor on the output node. The NAND gate is similar except *both* inputs must be (roughly speaking) above the threshold in order for the output to go low. If *either* input goes low, the output will charge toward the positive rail. Note also that if one input of the NAND gate is held high, the circuit behaves exactly as an inverter.

The axon circuit is formed by cascading multiple copies of this simple five transistor circuit in series. Let the voltage on the first capacitor of the $n^{th}$ stage be denoted by $u_n$ and the voltage on the second capacitor by $v_n$. Note that there is feedback from $u_{n+1}$ to the lower input of the NAND gate of the $n^{th}$ stage. Under quiescent conditions, the input to the first stage is low (at the negative rail), the $u$ nodes of all stages are high (at the positive rail), and all of the $v$ nodes are held low (at the negative rail). The feedback signal to the final stage in the line would be tied to the positive rail. The level of the bias voltages $\tau_1$ and $\tau_2$ determine whether or not a narrow pulse fed into the input of the first stage will propagate and, if so, the width and velocity with which it does.

In order to obtain a semi-quantitative understanding of how the axon circuit behaves, we will first consider the dynamics of a cascade of simple inverters (three sections of which are depicted in Figure 2) and then consider the addition of feedback. Under most circumstances, discharges will occur on a much faster time scale

than the recharges, so we make the simplifying assumption that when the input of an inverter reaches the threshold voltage, the output discharges instantaneously. Additionally, we assume that saturated transistors behave as ideal current sources (i.e., we neglect the Early effect) so that the recharges are linear ramps in time.

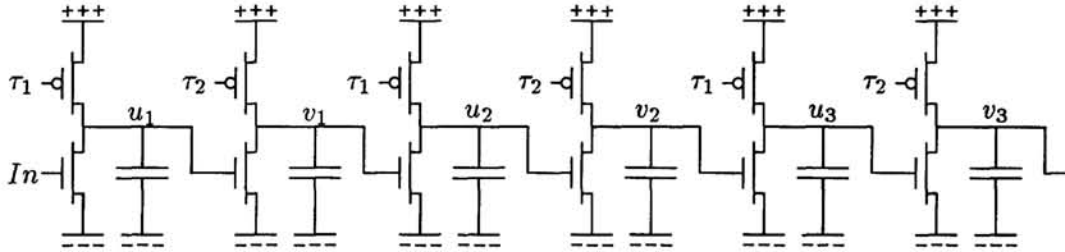

Figure 2: A cascade of pseudo-nMOS inverters.

Let $I_1$ and $I_2$ be the saturation currents in the pull-up transistors with bias voltages $\tau_1$ and $\tau_2$, respectively. Let $\Theta_1$ and $\Theta_2$ be the threshold voltages of the first and second inverters in a single stage, respectively. Also, let $\dot{u} = I_1/C$ and $\dot{v} = I_2/C$ be the slew rates for the $u$ and $v$ nodes, respectively. Let $\Delta_1 = \Theta_1/\dot{v}$ and $\Delta_2 = \Theta_2/\dot{u}$ be the time required for $u_n$ to charge from the negative rail up to $\Theta_1$ and for $v_n$ to charge from the negative rail up to $\Theta_2$, respectively. Finally, let $v_n^*$ denote the peak value attained by the $v$ signal of the $n^{th}$ stage.

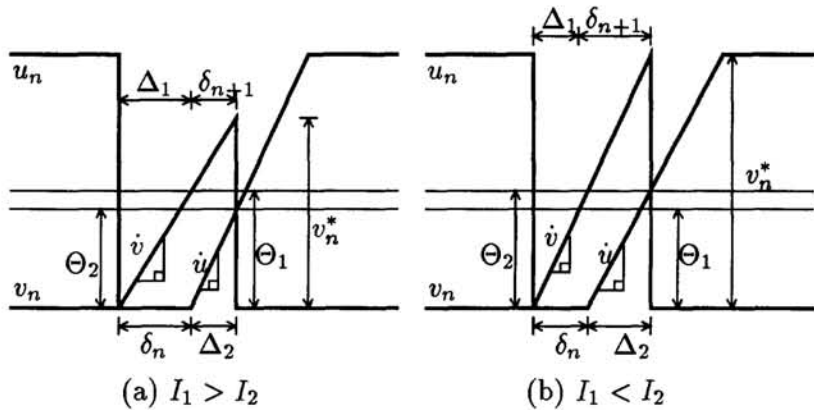

(a) $I_1 > I_2$                        (b) $I_1 < I_2$

Figure 3: Geometry of the idealized $u_n$ and $v_n$ signals under the bias conditions (a) $I_1 > I_2$ and (b) $I_1 < I_2$.

Consider what would happen if $v_{n-1}$ exceeded $\Theta_1$ for a time $\delta_n$. In this case, $u_n$ would be held low for $\delta_n$ and then released. Meanwhile, $v_n$ would ramp up to $\dot{v}\delta_n$. Then, $u_n$ would begin to charge toward the positive rail while $v_n$ continues to charge. This continues for a time $\Delta_2$ at which point $u_n$ will have reached $\Theta_2$ causing

$v_n$ to be discharged to the negative rail. Now, $u_{n+1}$ is held low while $v_n$ exceeds $\Theta_1$ this interval of time is precisely $\delta_{n+1}$. Figures 3a and 3b depict the geometry of the $u_n$ and $v_n$ signals in this scenario under the bias conditions $I_1 > I_2$ and $I_1 < I_2$, respectively. Simple Euclidean geometry implies that the evolution of $\delta_n$ will be governed by the first-order difference equation

$$\delta_{n+1} = \delta_n + (\Delta_2 - \Delta_1)$$

which is trivially solved by

$$\delta_n = \delta_1 + (n - 1)(\Delta_2 - \Delta_1).$$

Thus, the quantity $\Delta_2 - \Delta_1$ determines what happens to the width of the pulse as it propagates. In the event that $\Delta_2 < \Delta_1$, the pulse will shrink down to nothing from its initial width. If $\Delta_2 > \Delta_1$, the pulse width will grow without bound from its initial width. The pulse width is preserved only if $\Delta_2 = \Delta_1$. This last case, however, is unrealistic. There will always be component mismatches (with both systematic and random parts), which will cause the width of the pulse to grow and shrink as it propagates down the line, perhaps cancelling on average. Any systematic offsets will cause the pulse to shrink to nothing or to grow without bound as it propagates. In any event, information about the detailed timing of the initial pulse will have been completely lost.

Now, consider the action of the feedback in the axon circuit (Figure 1). If $u_n$ were to be held low for a time longer than $\Delta_1$ (i.e., the time it takes $v_n$ to charge up to $\Theta_1$), $u_{n+1}$ would come back and release $u_n$, regardless of the state of the input. Thus, the feedback enforces the condition $\delta_n \leq \Delta_1$. If $I_1 > I_2$ (i.e., $\Delta_2 < \Delta_1$), a pulse whose initial width is larger than $\Delta_1$ will be clipped to $\Delta_1$ and then shrink down to nothing and disappear. In the event that $I_1 < I_2$ (i.e., $\Delta_2 > \Delta_1$), a pulse whose initial duration is too small will grow up until its width is limited by the feedback. The axon circuit normally operates under the latter bias condition. The dynamics of the simple inverter chain cause a pulse which is to narrow to grow and the feedback loop serves to limit the pulse width; thus, the width of the pulse is restored in time. The feedback is also the source of the refractoriness in the axon; that is, until $u_{n+1}$ charges up to (roughly) $\Theta_1$, $v_{n-1}$ can have no effect on $u_n$.

## 3 EXPERIMENTAL DATA

In this section, data from a twenty-five stage axon will be shown. The chip was fabricated in a standard $2\mu m$ p-well (Orbit) CMOS process through MOSIS.

### Uniform Axon

A full space-time picture of pulse (taken at the $v$ nodes of the circuit) propagation down a uniform axon is depicted on the left in Figure 4. The graph on the right in Figure 4 shows the same data from a different perspective. The lower sloped curve represents the time of the initial rapid discharge of the $u$ node at each successive stage–this time marks the leading edge of the pulse taken at the $v$ node of that stage.

The upper sloped curve marks the time of the final rapid discharge of the $v$ node of each stage–this time is the end of the pulse taken at the $v$ node of that stage. The propagation velocity of the pulse is given (in units of *stages/s*) by the reciprocal of the slope of the lower inclined curve. The third curve is the difference of the other two and represents the pulse width as a function of position along the axon. The graph on the left of Figure 5 shows propagation velocity as a function of the $\tau_2$ bias voltage–so long as the pulse propagates, the velocity is nearly independent of $\tau_1$. Two orders of magnitude of velocity are shown in the plot; these are especially well matched to the time scales of motion in auditory and visual sensory data. The circuit is tunable over a much wider range of velocities (from about one stage per second to well in excess of $10^4 stages/s$). The graph on the right of Figure 5 shows pulse width as a function of $\tau_1$ for various values of $\tau_2$–the pulse width is mainly determined by $\tau_1$ with $\tau_2$ setting a lower limit.

**Tapered Axon**

In biological axons, the propagation velocity of an action potential is related to the diameter of the axon–the bigger the diameter, the greater the velocity. If the axon were tapered, the velocity of the action potential would change as it propagated. If the bias transistors in the axon circuit are operated in their subthreshold region, the effect of an exponentially tapered axon can be simulated by applying a small voltage difference to the ends of each of the $\tau_1$ and $\tau_2$ bias lines. (Lyon & Mead, 1989) These narrow wires are made with a relatively resistive layer (polysilicon); hence, putting a voltage difference across the ends will linearly interpolate the bias voltages for each stage along the line. In subthreshold, the bias currents are exponentially related to the bias voltages. Since the pulse width and velocity are related to the bias currents, we expect that a pulse will either speed up and get narrower or slow down and get wider (depending on the sign of the applied voltage) exponentially as a function of position along the line. The graph on the left of Figure 6 depicts the boundaries of a pulse as it propagates along of the axon circuit for a positive (*'s) and negative (x's) voltage difference applied to the $\tau$ lines. The graph on the right of Figure 6 shows the corresponding pulse width for each applied voltage difference. Note that in each case, the width changes by more than an order of magnitude, but the pulse maintains its integrity. That is, the pulse does not disappear nor does it split into multiple pulses–this behavior would not be possible if the pulse width were not restored in time.

## 4 CONCLUSIONS

In this paper we have presented a low-power, compact axon circuit, explained its operation, and presented data from a working chip fabricated through MOSIS. The circuit shares a number of features with its biological counterpart including an excitation threshold, a brief refractory period after pulse completion, pulse amplitude restoration, and pulse width restoration. It is tunable over orders of magnitude in pulse propagation velocity–including those well matched to the time scales of auditory and visual signals–and shows promise for use in synthetic neural systems which perform computations by correlating events which occur over both space and time such as those presented in (Horiuchi *et al*, 1991) and (Lazzaro & Mead, 1989).

## Acknowledgements

This material is based upon work supported in part under a National Science Foundation Graduate Research Fellowship, the Office of Naval Research, DARPA, and the Beckman Foundation.

## References

A. L. Hodgkin and A. K. Huxley, (1952). A Quantitative Description of Membrane Current and its Application to Conduction and Excitation in Nerve. *Journal of Physiology*, 117:6, 500-544.

T. Horiuchi, J. Lazzaro, A. Moore, and C. Koch, (1991). A Delay-Line Based Motion Detection Chip. *Advances in Neural Information Processing Systems 3.* San Mateo, CA: Morgan Kaufmann Publishers, Inc. 406-412.

J. Lazzaro and C. Mead, (1989). A Silicon Model of Auditory Localization. *Neural Computation*, 1:1, 47-57.

R. Lyon and C. Mead, (1989). Electronic Cochlea. *Analog VLSI and Neural Systems.* Reading, MA: Addison-Wesley Publishing Company, Inc. 279-302.

E. R. Lewis, (1968). Using Electronic Circuits to Model Simple Neuroelectric Interactions. *Proceedings of the IEEE*, 56:6, 931-949.

M. Mahowald and R. Douglas, (1991). A Silicon Neuron. *Nature*, 354:19, 515-518.

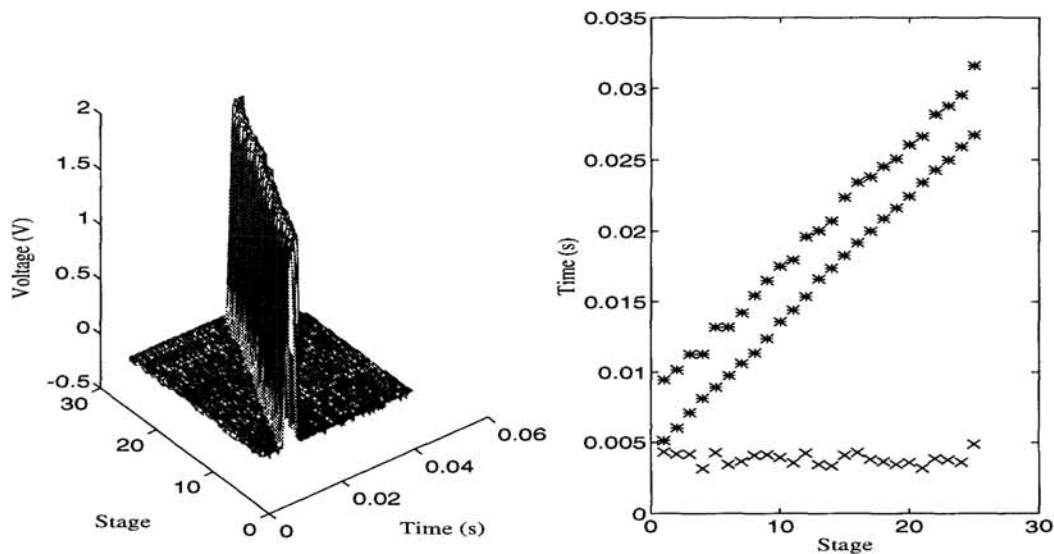

Figure 4: Pulse propagation along a uniform axon. (*Left*) Perspective view. (*Right*) Overhead view. *: pulse boundaries, x: pulse width. $\tau_1 = 0.720V$, $\tau_2 = 0.780V$. $Velocity = 1,100 stages/s$, $Width = 3.8ms$

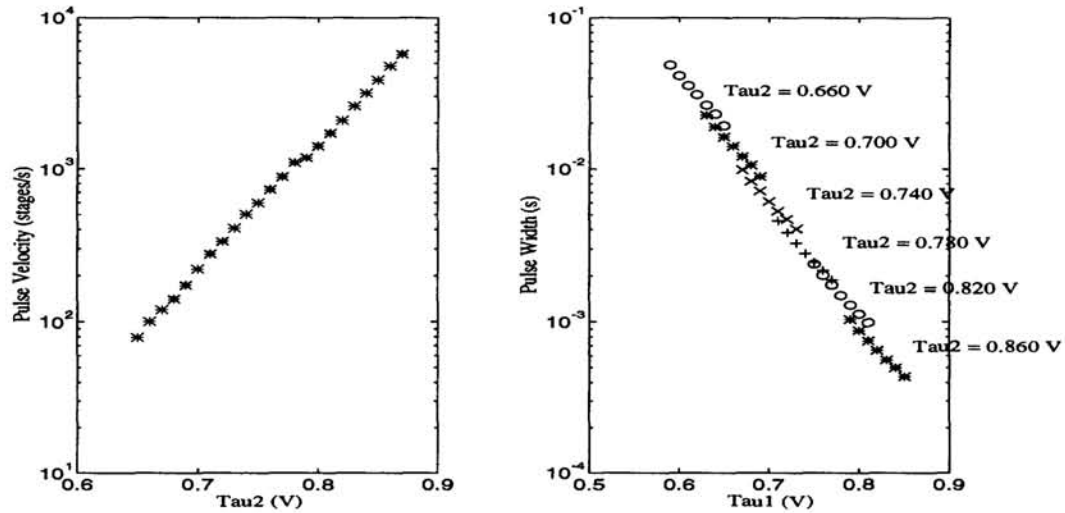

Figure 5: Uniform axon. (*Left*) Pulse velocity as a function of $\tau_2$. (*Right*) Pulse width as a function of $\tau_1$ for various values of $\tau_2$.

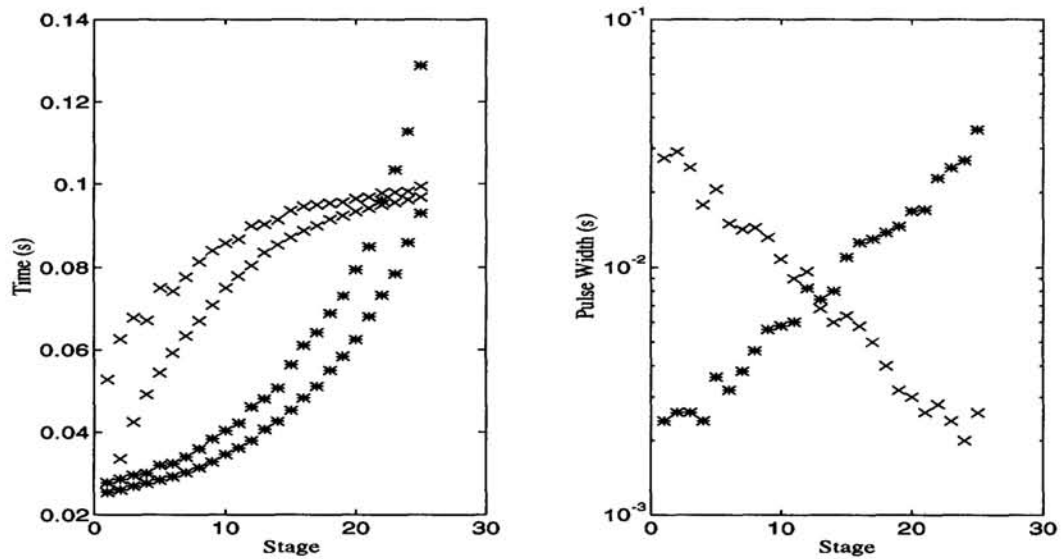

Figure 6: (*Left*) Pulse propagation along a tapered axon. (*Right*) Pulse width as a function of position along a tapered axon. $*$: $\tau_1^{left} = 0.770V$, $\tau_1^{right} = 0.600V$, $\tau_2^{left} = 0.820V$, $\tau_2^{right} = 0.650$. x: $\tau_1^{left} = 0.600V$, $\tau_1^{right} = 0.770V$, $\tau_2^{left} = 0.650V$, $\tau_2^{right} = 0.820V$.